# A Realizable Learning Task which Exhibits Overfitting

**Siegfried Bös**
Laboratory for Information Representation, RIKEN,
Hirosawa 2-1, Wako-shi, Saitama, 351-01, Japan
email: boes@zoo.riken.go.jp

## Abstract

In this paper we examine a perceptron learning task. The task is realizable since it is provided by another perceptron with identical architecture. Both perceptrons have nonlinear sigmoid output functions. The gain of the output function determines the level of nonlinearity of the learning task. It is observed that a high level of nonlinearity leads to overfitting. We give an explanation for this rather surprising observation and develop a method to avoid the overfitting. This method has two possible interpretations, one is learning with noise, the other cross–validated early stopping.

## 1 Learning Rules from Examples

The property which makes feedforward neural nets interesting for many practical applications is their ability to approximate functions, which are given only by examples. Feed-forward networks with at least one hidden layer of nonlinear units are able to approximate each continuous function on a $N$-dimensional hypercube arbitrarily well. While the existence of neural function approximators is already established, there is still a lack of knowledge about their practical realizations. Also major problems, which complicate a good realization, like overfitting, need a better understanding.

In this work we study overfitting in a one–layer perceptron model. The model allows a good theoretical description while it exhibits already a qualitatively similar behavior as the multilayer perceptron.

A one–layer perceptron has $N$ input units and one output unit. Between input and output it has one layer of adjustable weights $W_i, (i = 1, \ldots, N)$. The output $z$ is a possibly nonlinear function of the weighted sum of inputs $x_i$, i.e.

$$z = g(h), \quad \text{with} \quad h = \frac{1}{\sqrt{N}} \sum_{i=1}^{N} W_i x_i. \tag{1}$$

The quality of the function approximation is measured by the difference between the correct output $z_*$ and the net's output $z$ averaged over all possible inputs. In the *supervised learning* scheme one trains the network using a set of examples $\underline{x}^\mu$ ($\mu = 1, \ldots, P$), for which the correct output is known. It is the learning task to minimize a certain cost function, which measures the difference between the correct output $z_*^\mu$ and the net's output $z^\mu$ averaged over all examples.

Using the mean squared error as a suitable measure for the difference between the outputs, we can define the *training error* $E_T$ and the *generalization error* $E_G$ as

$$E_T := \frac{1}{2P} \sum_{\mu=1}^{P} [z_*(\underline{x}^\mu) - z(\underline{x}^\mu)]^2, \qquad E_G := \frac{1}{2} < [z_*(\underline{x}) - z(\underline{x})]^2 >_{\{\underline{x}\}} . \quad (2)$$

The development of both errors as a function of the number $P$ of trained examples is given by the *learning curves*. Training is conventionally done by gradient descend.

For theoretical purposes it is very useful to study learning tasks, which are provided by a second network, the so–called *teacher network*. This concept allows a more transparent definition of the difficulty of the learning task. Also the monitoring of the training process becomes clearer, since it is always possible to compare the student network and the teacher network directly.

Suitable quantities for such a comparison are, in the perceptron case, the following *order parameters*,

$$r := \frac{1}{\|\underline{W}\|} \sum_{i=1}^{N} W_i^* W_i, \qquad q := \|\underline{W}\| = \sqrt{\sum_{i=1}^{N} (W_i)^2} . \quad (3)$$

Both have a very transparent interpretation, $r$ is the normalized overlap between the weight vectors of teacher and student, and $q$ is the norm of the student's weight vector. These order parameters can also be used in multilayer learning, but their number increases with the number of all possible permutations between the hidden units of teacher and student.

## 2   The Learning Task

Here we concentrate on the case in which a student perceptron has to learn a mapping provided by another perceptron. We choose identical networks for teacher and student. Both have the same sigmoid output function, i.e. $g_*(h) = g(h) = \tanh(\gamma h)$. Identical network architectures of teacher and student are *realizable* tasks. In principle the student is able to learn the task provided by the teacher exactly. *Unrealizable* tasks can not be learnt exactly, there remains always a finite error.

If we use uniformally distributed random inputs $\underline{x}$ and weights $\underline{W}$, the weighted sum $h$ in (1) can be assumed as Gaussian distributed. Then we can express the generalization error (2) by the order parameters (3),

$$E_G = \int Dz_1 \int Dz_2 \frac{1}{2} \left\{ \tanh[\gamma z_1] - \tanh\left[q(r z_1 + \sqrt{1 - r^2} z_2)\right] \right\}^2 , \quad (4)$$

with the Gaussian measure

$$\int Dz := \int_{-\infty}^{+\infty} \frac{dz}{\sqrt{2\pi}} \exp\left(-\frac{z^2}{2}\right) . \quad (5)$$

From equation (4) we can see how the student learns the gain $\gamma$ of the teachers output function. It adjusts the norm $q$ of its weights. The gain $\gamma$ plays an important role since it allows to tune the function $\tanh(\gamma h)$ between a linear function ($\gamma \ll 1$) and a highly nonlinear function ($\gamma \gg 1$). Now we want to determine the learning curves of this task.

## 3   Emergence of Overfitting

### 3.1   Explicit Expression for the Weights

Below the storage capacity of the perceptron, i.e. $\alpha = 1$, the minimum of the training error $E_T$ is zero. A zero training error implies that every example has been learnt exactly, thus

$$E_T = 0 \quad \Rightarrow \quad z^\mu = z_*^\mu \quad \Rightarrow \quad h^\mu = g^{-1}\left(g_*(h_*^\mu)\right) = h_*^\mu. \tag{6}$$

The weights with minimal norm that fulfill this condition are given by the *Pseudoinverse* (see Hertz *et al.* 1991),

$$W_i = \sum_{\mu,\nu=1}^{P} h_*^\mu \, (C^{-1})_{\mu\nu} \, x_i^\nu, \qquad C_{\mu\nu} = \frac{1}{N} \sum_{i=1}^{N} x_i^\mu x_i^\nu. \tag{7}$$

Note, that the weights are completely independent of the output function $g(h) = g_*(h)$. They are the same as in the simplest realizable case, linear perceptron learns linear perceptron.

### 3.2   Statistical Mechanics

The calculation of the order parameters can be done by a method from statistical mechanics which applies the commonly used *replica method*. For details about the replica approach see Hertz *et al.* (1991). The solution of the continuous perceptron problem can be found in Bös *et al.* (1993). Since the results of the statistical mechanics calculations are exact only in the thermodynamic limit, i.e. $N \to \infty$, the variable $\alpha$ is the more natural measure. It is defined as the fraction of the number of patterns $P$ over the system size $N$, i.e. $\alpha := P/N$. In the thermodynamic limit $N$ and $P$ are infinite, but $\alpha$ is still finite. Normally, reasonable system sizes, such as $N \geq 100$, are already well described by this theory.

Usually one concentrates on the zero temperature limit, because this implies that the training error $E_T$ accepts its absolute minimum for every number of presented examples $P$. The corresponding order parameters for the case, linear perceptron learns linear student, are

$$q = \gamma\sqrt{\alpha}, \qquad r = \sqrt{\alpha}. \tag{8}$$

The zero temperature limit can also be called *exhaustive training*, since the student net is trained until the absolute minimum of $E_T$ is reached.

For small $\alpha$ and high gains $\gamma$, i.e levels of nonlinearity, exhaustive training leads to overfitting. That means the generalization error $E_G(\alpha)$ is not, as it should, monotonously decreasing with $\alpha$. It is one reason for overfitting, that the training follows too strongly the examples. The *critical gain* $\gamma_c$, which determines whether the generalization error $E_G(\alpha)$ is increasing or decreasing function for small values of $\alpha$, can be determined by a linear approximation. For small $\alpha$, both order parameters (3) are small, and the student's tanh–function in (4) can be approximated by a linear function. This simplifies the equation (4) to the following expression,

$$E_G(\epsilon) = E_G(0) - \frac{\epsilon}{2}\left[2H(\gamma) - \gamma\right], \quad \text{with} \quad H(\gamma) := \int Dz \, \tanh(\gamma z) \, z. \tag{9}$$

Since the function $H(\gamma)$ has an upper bound, i.e. $\sqrt{2/\pi}$, the critical gain is reached if $\gamma_c = 2H(\gamma_c)$. The numerical solution gives $\gamma_c = 1.3371$. If $\gamma$ is higher, the slope of $E_G(\alpha)$ is positive for small $\alpha$. In the following considerations we will use always the gain $\gamma = 5$ as an example, since this is an intermediate level of nonlinearity.

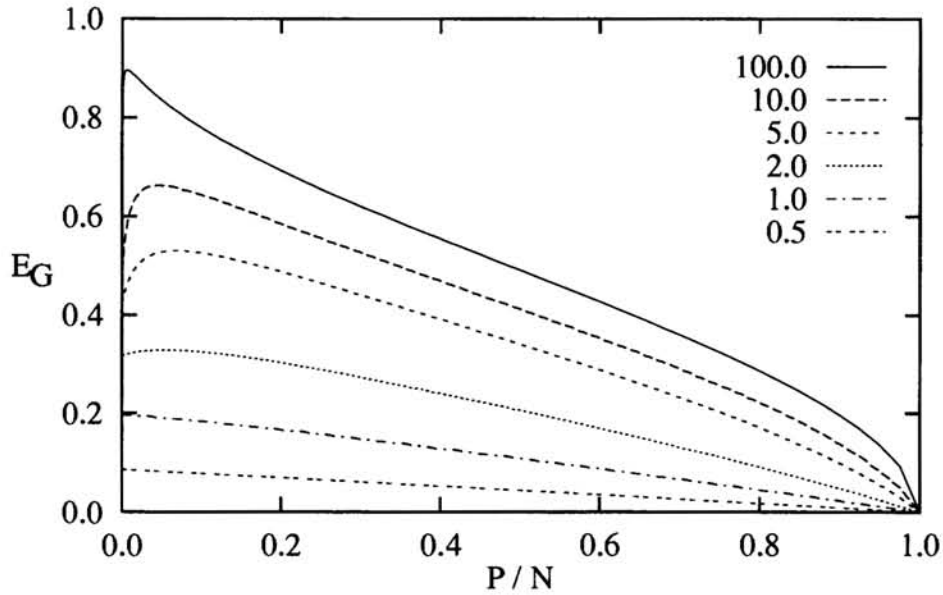

Figure 1: Learning curves $E(\alpha)$ for the problem, tanh–perceptron learns tanh–perceptron, for different values of the gain $\gamma$. Even in this realizable case, exhaustive training can lead to overfitting, if the gain $\gamma$ is high enough.

### 3.3 How to Understand the Emergence of Overfitting

Here the evaluation of the generalization error in dependence of the order parameters $r$ and $q$ is helpful. Fig. 2 shows the function $E_G(r,q)$ for $r$ between 0 and 1 and $q$ between 0 and $1.2\gamma$.

The exhaustive training in realizable cases follows always the line $q(r) = \gamma r$ independent of the actual output function. That means, training is guided only by the training error and not by the generalization error. If the gain $\gamma$ is higher than $\gamma_c$, the line $E_G = E_G(0,0)$ starts with a lower slope than $q(r) = \gamma r$, which results in overfitting.

## 4  How to Avoid Overfitting

From Fig. 2 we can guess already that $q$ increases too fast compared to $r$. Maybe the ratio between $q$ and $r$ is better during the training process. So we have to develop a description for the training process first.

### 4.1 Training Process

We found already that the order parameters for finite temperatures $(T > 0)$ of the statistical mechanics approach are a good description of the training process in an unrealizable learning task (Bös 1995). So we use the finite temperature order parameters also in this task. These are, again taken from the task 'linear perceptron learns linear perceptron',

$$q(\alpha,a) = \gamma \sqrt{\left(\frac{\alpha}{a}\right)\frac{(1+\alpha)\,a - 2\alpha}{a^2 - \alpha}}, \quad r(\alpha,a) = \sqrt{\left(\frac{\alpha}{a}\right)\frac{a^2 - \alpha}{(1+\alpha)\,a - 2\alpha}}, \quad (10)$$

with the temperature dependent variable

$$a := 1 + [\beta\,(Q - q)]^{-1} . \quad (11)$$

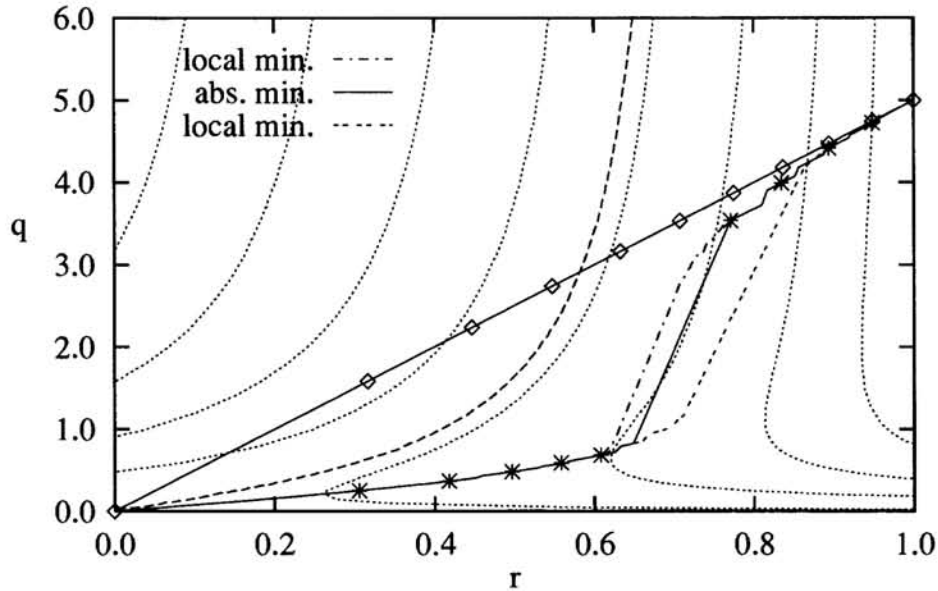

Figure 2: Contour plot of $E_G(r,q)$ defined by (4), the generalization error as a function of the two order parameters. Starting from the minimum $E_G = 0$ at $(r,q) = (1,5)$ the contour lines for $E_G = 0.1, 0.2, ..., 0.8$ are given (dotted lines). The dashed line corresponds to $E_G(0,0) = 0.42$. The solid lines are parametric curves of the order parameters $(r,q)$ for certain training strategies. The straight line illustrates exhaustive training, the lower ones the optimal training, which will be explained in Fig. 3. Here the gain $\gamma = 5$.

The zero temperature limit corresponds to $a = 1$. We will show now that the decrease of the temperature dependent parameter $a$ from $\infty$ to 1, describes the evolution of the order parameters during the training process. In the training process the natural parameter is the number of parallel training steps $t$. In each parallel training step all patterns are presented once and all weights are updated. Fig. 3 shows the evolution of the order parameters (10) as parametric curves $(r,q)$.

The exhaustive learning curve is defined by $a = 1$ with the parameter $\alpha$ (solid line). For each $\alpha$ the training ends on this curve. The dotted lines illustrate the training process, $a$ runs from infinity to 1. Simulations of the training process have shown that this theoretical curve is a good description, at least after some training steps. We will now use this description of the training process for the definition of an optimized training strategy.

## 4.2   Optimal temperature

The optimized training strategy chooses not $a = 1$ or the corresponding temperature $T = 0$, but the value of $a$ (i.e. temperature), which minimizes the generalization error $E_G$. In the lower solid curve indicating the parametric curve $(r,q)$ the value of $a$ is chosen for every $\alpha$, which minimizes $E_G$. The function $E_G(a)$ has two minima between $\alpha = 0.5$ and 0.7. The solid line indicates always the absolute minimum. The parametric curves corresponding to the local minima are given by the double dashed and dash–dotted lines. Note, that the optimized value $a$ is always related to an optimized temperature through equation (11). But the parameter $a$ is also related to the number of training steps $t$.

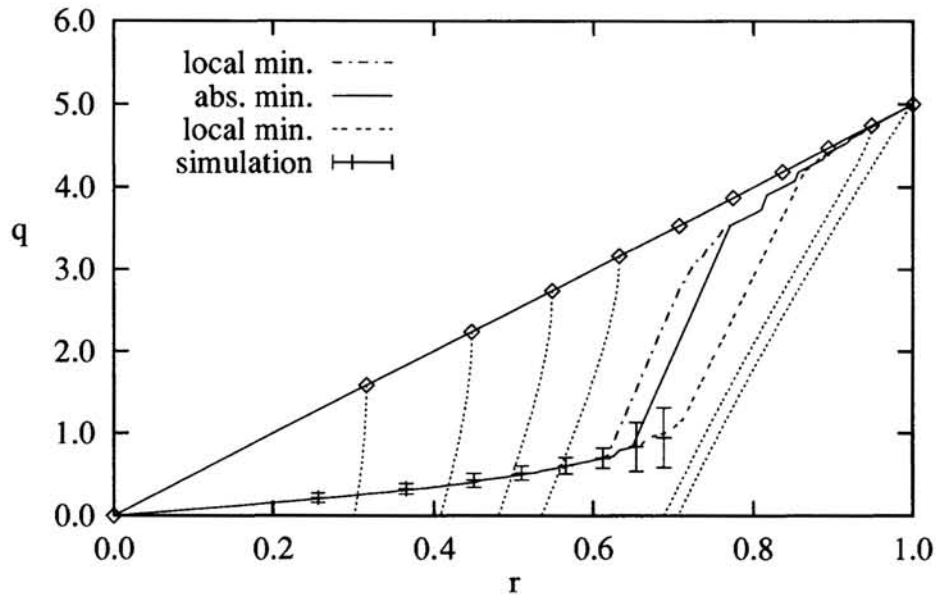

Figure 3: Training process. The order parameters (10) as parametric curves $(r, q)$ with the parameters $\alpha$ and $a$. The straight solid line corresponds to exhaustive learning, i.e. $a = 1$ (marks at $\alpha = 0.1, 0.2, \ldots 1.0$). The dotted lines describe the training process for fixed $\alpha$. Iterative training reduces the parameter $a$ from $\infty$ to 1. Examples for $\alpha = 0.1, 0.2, 0.3, 0.4, 0.9, 0.99$ are given. The lower solid line is an optimized learning curve. To achieve this curve the value of $a$ is chosen, which minimizes $E_G$ absolutely. Between $\alpha \simeq 0.5$ and 0.7 the error $E_G$ has two minima; the double–dashed and dash–dotted lines indicate the second, local minimum of $E_G$. Compare with Fig. 2, to see which is the absolute and which the local minimum of $E_G$. A naive early stopping procedure ends always in the minimum with the smaller $q$, since it is the first minimum during the training process (see simulation indicated with errorbars).

## 4.3  Early Stopping

Fig. 3 and Fig. 2 together indicate that an earlier stopping of the training process can avoid the overfitting. But in order to determine the stopping point one has to know the actual generalization error during the training. *Cross–validation* tries to provide an approximation for the real generalization error. The cross–validation error $E_{CV}$ is defined like $E_T$, see (2), on a set of examples, which are not used during the training. Here we calculate the optimum using the real generalization error, given by $r$ and $q$, to determine the optimal point for early stopping. It is a lower bound for training with finite cross–validation sets. Some preliminary tests have shown that already small cross–validation sets approximate the real $E_G$ quite well. Training is stopped, when $E_G$ increases. The resulting curve is given by the errorbars in Fig. 3. The errorbars indicate the standard deviation of a simulation with $N = 100$ averaged over 50 trials.

In Fig. 4 the same results are shown as learning curves $E_G(\alpha)$. There one can see clearly that the early stopping strategy avoids the overfitting.

## 5  Summary and outlook

In this paper we have shown that overfitting can also emerge in realizable learning tasks. The calculation of a critical gain and the contour lines in Fig. 2 imply, that

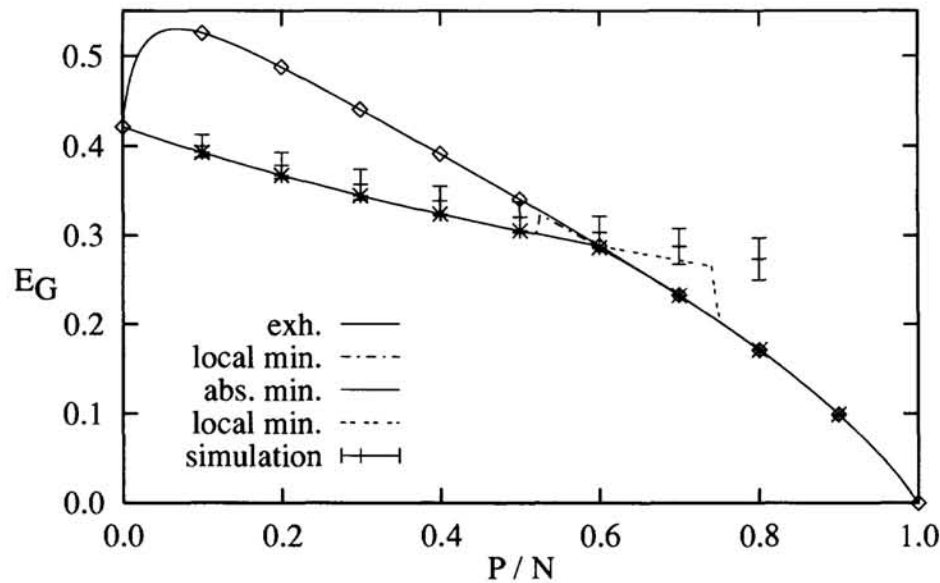

Figure 4: Learning curves corresponding to the parametric curves in Fig. 3. The upper solid line shows again exhaustive training. The optimized finite temperature curve is the lower solid line. From $\alpha = 0.6$ exhaustive and optimal training lead to identical results (see marks). The simulation for early stopping (errorbars) finds the first minimum of $E_G$.

the reason for the overfitting is the nonlinearity of the problem. The network adjusts slowly to the nonlinearity of the task. We have developed a method to avoid the overfitting, it can be interpreted in two ways.

Training at a finite temperature reduces overfitting. It can be realized, if one trains with noisy examples. In the other interpretation one learns without noise, but stops the training earlier. The early stopping is guided by cross–validation. It was observed that early stopping is not completely simple, since it can lead to a local minimum of the generalization error. One should be aware of this possibility, before one applies early stopping.

Since multilayer perceptrons are built of nonlinear perceptrons, the same effects are important for multilayer learning. A study with large scale simulations (Müller et al. 1995) has shown that overfitting occurs also in realizable multilayer learning tasks.

### Acknowledgments
I would like to thank S. Amari and M. Opper for stimulating discussions, and M. Herrmann for hints concerning the presentation.

### References
S. Bös. (1995) Avoiding overfitting by finite temperature learning and cross–validation. *International Conference on Artificial Neural Networks'95* Vol.2, p.111.
S. Bös, W. Kinzel & M. Opper. (1993) Generalization ability of perceptrons with continuous outputs. *Phys. Rev. E* **47**:1384–1391.
J. Hertz, A. Krogh & R. G. Palmer. (1991) *Introduction to the Theory of Neural Computation.* Reading: Addison–Wesley.
K. R. Müller, M. Finke, N. Murata, K. Schulten & S. Amari. (1995) On large scale simulations for learning curves, *Neural Computation* in press.